# Correlation Coefficients Are Insufficient for Analyzing Spike Count Dependencies

**Arno Onken**
Technische Universität Berlin / BCCN Berlin
Franklinstr. 28/29, 10587 Berlin, Germany
`aonken@cs.tu-berlin.de`

**Steffen Grünewälder**
University College London
Gower Street, London WC1E 6BT, UK
`steffen@cs.ucl.ac.uk`

**Klaus Obermayer**
Technische Universität Berlin / BCCN Berlin
`oby@cs.tu-berlin.de`

## Abstract

The linear correlation coefficient is typically used to characterize and analyze dependencies of neural spike counts. Here, we show that the correlation coefficient is in general insufficient to characterize these dependencies. We construct two neuron spike count models with Poisson-like marginals and vary their dependence structure using copulas. To this end, we construct a copula that allows to keep the spike counts uncorrelated while varying their dependence strength. Moreover, we employ a network of leaky integrate-and-fire neurons to investigate whether weakly correlated spike counts with strong dependencies are likely to occur in real networks. We find that the entropy of uncorrelated but dependent spike count distributions can deviate from the corresponding distribution with independent components by more than 25 % and that weakly correlated but strongly dependent spike counts are very likely to occur in biological networks. Finally, we introduce a test for deciding whether the dependence structure of distributions with Poisson-like marginals is well characterized by the linear correlation coefficient and verify it for different copula-based models.

## 1 Introduction

The linear correlation coefficient is of central importance in many studies that deal with spike count data of neural populations. For example, a low correlation coefficient is often used as an evidence for independence in recorded data and to justify simplifying model assumptions (e.g. [1, 2]). In line with this many computational studies constructed distributions for observed data based solely on reported correlation coefficients [3, 4, 5, 6]. The correlation coefficient is in this sense treated as an equivalent to the full dependence.

The correlation coefficient is also extensively used in combination with information measures such as the Fisher information (for continuous variables only) and the Shannon information to assess the importance of couplings between neurons for neural coding [7]. The discussion in the literature encircles two main topics. On the one hand, it is debated whether pairwise correlations versus higher order correlations across different neurons are sufficient for obtaining good estimates of the information (see e.g. [8, 9, 10]). On the other hand, it is questioned whether correlations matter at all (see e.g. [11, 12, 13]). In [13], for example, based on the correlation coefficient it was argued that the impact of correlations is negligible for small populations of neurons.

The correlation coefficient is one measure of dependence among others. It has become common to report only the correlation coefficient of recorded spike trains without reporting any other properties

of the actual dependence structure (see e.g. [3, 14, 15]). The problem with this common practice is that it is unclear beforehand whether the linear correlation coefficient suffices to describe the dependence or at least the *relevant* part of the dependence. Of course, it is well known that *uncorrelated* does not imply *statistically independent*. Yet, it might seem likely that this is not important for realistic spike count distributions which have a Poisson-like shape. Problems could be restricted to pathological cases that are very unlikely to occur in realistic biological networks. At least one might expect to find a tendency of weak dependencies for uncorrelated distributions with Poisson-like marginals. It might also seem likely that these dependencies are unimportant in terms of typical information measures even if they are present and go unnoticed or are ignored.

In this paper we show that these assumptions are false. Indeed, the dependence structure can have a profound impact on the information of spike count distributions with Poisson-like single neuron statistics. This impact can be substantial not only for large networks of neurons but even for two neuron distributions. As a matter of fact, the correlation coefficient places only a weak constraint on the dependence structure. Moreover, we show that uncorrelated or weakly correlated spike counts with strong dependencies are very likely to be common in biological networks. Thus, it is not sufficient to report only the correlation coefficient or to derive strong implications like independence from a low correlation coefficient alone. At least a statistical test should be applied that states for a given significance level whether the dependence is well characterized by the linear correlation coefficient. We will introduce such a test in this paper. The test is adjusted to the setting that a neuroscientist typically faces, namely the case of Poisson-like spike count distributions of single neurons and small numbers of samples.

In the next section, we describe state-of-the-art methods for modeling dependent spike counts, to compute their entropy, and to generate network models based on integrate-and-fire neurons. Section 3 shows examples of what can go wrong for entropy estimation when relying on the correlation coefficient only. Emergences of such cases in simple network models are explored. Section 4 introduces the linear correlation test which is tailored to the needs of neuroscience applications and the section examines its performance on different dependence structures. The paper concludes with a discussion of the advantages and limitations of the presented methods and cases.

## 2  General methods

We will now describe formal aspects of spike count models and their Shannon information.

### 2.1  Copula-based models with discrete marginals

A copula is a cumulative distribution function (CDF) which is defined on the unit hypercube and has uniform marginals [16]. Formally, a bivariate copula $C$ is defined as follows:

**Definition 1.** *A copula is a function* $C : [0, 1]^2 \longrightarrow [0, 1]$ *such that:*

1. $\forall u, v \in [0, 1]$: $C(u, 0) = 0 = C(0, v)$ *and* $C(u, 1) = u$ *and* $C(1, v) = v$.

2. $\forall u_1, v_1, u_2, v_2 \in [0, 1]$ *with* $u_1 \leq u_2$ *and* $v_1 \leq v_2$:
   $C(u_2, v_2) - C(u_2, v_1) - C(u_1, v_2) + C(u_1, v_1) \geq 0$.

Copulas can be used to couple arbitrary marginal CDF's $F_{X_1}, F_{X_2}$ to form a joint CDF $F_{\vec{X}}$, such that $F_{\vec{X}}(r_1, r_2) = C(F_{X_1}(r_1), F_{X_2}(r_2))$ holds [16]. There are many families of copulas representing different dependence structures. One example is the bivariate Frank family [17]. Its CDF is given by

$$C_\theta(u, v) = \begin{cases} -\frac{1}{\theta} \ln\left(1 + \frac{(e^{-\theta u} - 1)(e^{-\theta v} - 1)}{e^{-\theta} - 1}\right) & \text{if } \theta \neq 0, \\ uv & \text{if } \theta = 0. \end{cases} \tag{1}$$

The Frank family is commutative and radial symmetric: its probability density $c_\theta$ abides by $\forall (u, v) \in [0, 1]^2 : c_\theta(u, v) = c_\theta(1 - u, 1 - v)$ [17]. The scalar parameter $\theta$ controls the strength of dependence. As $\theta \to \pm\infty$ the copula approaches deterministic positive/negative dependence: knowledge of one variable implies knowledge of the other (so-called Fréchet-Hoeffding bounds [16]). The linear correlation coefficient is capable of measuring this dependence. Another example is the bivariate Gaussian copula family defined as $C_\theta(u, v) = \phi_\theta(\phi^{-1}(u), \phi^{-1}(v))$, where $\phi_\theta$ is the CDF of

the bivariate zero-mean unit-variance multivariate normal distribution with correlation $\theta$ and $\phi^{-1}$ is the inverse of the CDF of the univariate zero-mean unit-variance Gaussian distribution. This family can be used to construct multivariate distributions with Gauss-like dependencies and arbitrary marginals.

For a given realization $\vec{r}$, which can represent the counts of two neurons, we can set $u_i = F_{X_i}(r_i)$ and $F_X(\vec{r}) = C_\theta(\vec{u})$, where $F_{X_i}$ can be arbitrary univariate CDF's. Thereby, we can generate a multivariate distribution with specific marginals $F_{X_i}$ and a dependence structure determined by $C$.

Copulas allow us to have different discrete marginal distributions [18, 19]. Typically, the Poisson distribution is a good approximation to spike count variations of single neurons [20]. For this distribution the CDF's of the marginals take the form

$$F_{X_i}(r; \lambda_i) = \sum_{k=0}^{\lfloor r \rfloor} \frac{\lambda_i^k}{k!} e^{-\lambda_i},$$

where $\lambda_i$ is the mean spike count of neuron $i$ for a given bin size. We will also use the negative binomial distribution as a generalization of the Poisson distribution:

$$F_{X_i}(r; \lambda_i, \upsilon_i) = \sum_{k=0}^{\lfloor r \rfloor} \frac{\lambda_i^k}{k!} \frac{1}{(1 + \frac{\lambda_i}{\upsilon_i})^{\upsilon_i}} \frac{\Gamma(\upsilon_i + k)}{\Gamma(\upsilon_i)(\upsilon_i + \lambda_i)^k},$$

where $\Gamma$ is the gamma function. The additional parameter $\upsilon_i$ controls the degree of overdispersion: the smaller the value of $\upsilon_i$, the greater the Fano factor: the variance is given by $\lambda_i + \frac{\lambda_i^2}{\upsilon_i}$. As $\upsilon_i$ approaches infinity, the negative binomial distribution converges to the Poisson distribution.

Likelihoods of discrete vectors can be computed by applying the inclusion-exclusion principle of Poincaré and Sylvester. The probability of a realization $(x_1, x_2)$ is given by $P_{\vec{X}}(x_1, x_2) = F_{\vec{X}}(x_1, x_2) - F_{\vec{X}}(x_1 - 1, x_2) - F_{\vec{X}}(x_1, x_2 - 1) + F_{\vec{X}}(x_1 - 1, x_2 - 1)$. Thus, we can compute the probability mass of a realization $\vec{x}$ using only the CDF of $\vec{X}$.

## 2.2 Computation of information entropy

The Shannon entropy [21] of dependent spike counts $\vec{X}$ is a measure of the information that a decoder is missing when it does not know the value $\vec{x}$ of $\vec{X}$. It is given by

$$H(\vec{X}) = \mathbb{E}[I(\vec{X})] = \sum_{\vec{x} \in \mathbb{N}^d} P_{\vec{X}}(\vec{x}) I(\vec{x}),$$

where $I(\vec{x}) = -\log_2(P_{\vec{X}}(\vec{x}))$ is the self-information of the realization $\vec{x}$.

## 2.3 Leaky integrate-and-fire model

The leaky integrate-and-fire neuron is a simple neuron model that models only subthreshold membrane potentials. The equation for the membrane potential is given by

$$\tau_m \frac{dV}{dt} = E_L - V + R_m I_s,$$

where $E_L$ denotes the resting membrane potential, $R_m$ is the total membrane resistance, $I_s$ is the synaptic input current, and $\tau_m$ is the time constant. The model is completed by a rule which states that whenever $V$ reaches a threshold $V_{th}$, an action potential is fired and $V$ is reset to $V_{reset}$ [22]. In all of our simulations we used $\tau_m = 20\,\text{ms}$, $R_m = 20\,\text{M}\Omega$, $V_{th} = -50\,\text{mV}$, and $V_{reset} = V_{init} = -65\,\text{mV}$, which are typical values found in [22]. Current-based synaptic input for an isolated presynaptic release that occurs at time $t = 0$ can be modeled by the so-called $\alpha$-function [22]: $I_s = I_{max} \frac{t}{\tau_s} \exp(1 - \frac{t}{\tau_s})$. The function reaches its peak $I_s$ at time $t = \tau_s$ and then decays with time constant $\tau_s$. We can model an excitatory synapse by a positive $I_{max}$ and an inhibitory synapse by a negative $I_{max}$. We used $I_{max} = 1\,\text{nA}$ for excitatory synapses, $I_{max} = -1\,\text{nA}$ for inhibitory synapses, and $\tau_s = 5\,\text{ms}$.

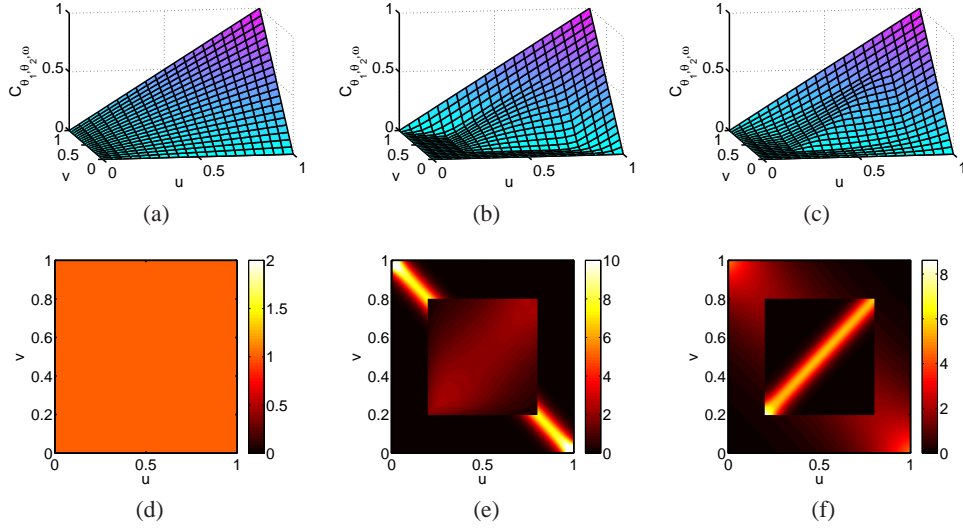

Figure 1: Cumulative distribution functions (a-c) and probability density functions (d-f) of selected Frank shuffle copulas. (a, d): Independence: $\theta_1 = \theta_2 = 0$. (b, e): Strong negative dependence in outer square: $\theta_1 = -30, \theta_2 = 5, \omega = 0.2$. (c, f): Strong positive dependence in inner square: $\theta_1 = -5, \theta_2 = 30, \omega = 0.2$.

## 3 Counter examples

In this section we describe entropy variations that can occur when relying on the correlation coefficient only. We will evaluate this effect for models of spike counts which have Poisson-like marginals and show that such effects can occur in very simple biological networks.

### 3.1 Frank shuffle copula

We will now introduce the Frank shuffle copula family. This copula family allows arbitrarily strong dependencies with a correlation coefficient of zero for attached Poisson-like marginals. It uses two Frank copulas (see Section 2.1) in different regions of its domain such that the linear correlation coefficient would vanish.

**Proposition 1.** *The following function defines a copula* $\forall \theta_1, \theta_2 \in \mathbb{R}$, $\omega \in [0, 0.5]$ :

$$C_{\theta_1,\theta_2,\omega}(u,v) = \begin{cases} C_{\theta_1}(u,v) - \varsigma_{\theta_1}(\omega,\omega,u,v) + z_{\theta_1,\theta_2,\omega}(min\{u,v\})\varsigma_{\theta_2}(\omega,\omega,u,v) & \text{if } (u,v) \in \\ & (\omega, 1-\omega)^2, \\ C_{\theta_1}(u,v) & \text{otherwise,} \end{cases}$$

*where* $\varsigma_\theta(u_1,v_1,u_2,v_2) = C_\theta(u_2,v_2) - C_\theta(u_2,v_1) - C_\theta(u_1,v_2) + C_\theta(u_1,v_1)$ *and* $z_{\theta_1,\theta_2,\omega}(m) = \varsigma_{\theta_1}(\omega,\omega,m,1-\omega)/\varsigma_{\theta_2}(\omega,\omega,m,1-\omega)$.

The proof of the copula properties is given in Appendix A. This family is capable of modeling a continuum between independence and deterministic dependence while keeping the correlation coefficient at zero. There are two regions: the outer region $[0,1]^2 \setminus (\omega, 1-\omega)^2$ contains a Frank copula with $\theta_1$ and the inner square $(\omega, 1-\omega)^2$ contains a Frank copula with $\theta_2$ modified by a factor $z$. If we would restrict our analysis to copula-based distributions with continuous marginals it would be sufficient to select $\theta_1 = -\theta_2$ and to adjust $\omega$ such that the correlation coefficient would vanish. In such cases, the factor $z$ would be unnecessary. For discrete marginals, however, this is not sufficient as the CDF is no longer a continuous function of $\omega$. Different copulas of this family are shown in Fig. 1.

We will now investigate the impact of this dependence structure on the entropy of copula-based distributions with Poisson-like marginals while keeping the correlation coefficient at zero. Introducing more structure into a distribution typically reduces its entropy. Therefore, we expect that the entropy can vary considerably for different dependence strengths, even though the correlation is always zero.

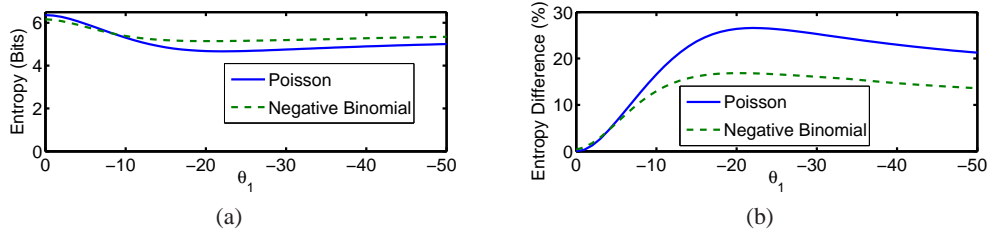

(a)                                                       (b)

Figure 2: Entropy of distributions based on the Frank shuffle copula $C_{\theta_1,\theta_2,\omega}$ for $\omega = 0.05$ and different dependence strengths $\theta_1$. The second parameter $\theta_2$ was selected such that the absolute correlation coefficient was below $10^{-10}$. For Poisson marginals, we selected rates $\lambda_1 = \lambda_2 = 5$. For $100$ ms bins this would correspond to firing rates of $50$ Hz. For negative binomial marginals we selected rates $\lambda_1 = 2.22$, $\lambda_2 = 4.57$ and variances $\sigma_1^2 = 4.24$, $\sigma_2^2 = 10.99$ (values taken from experimental data recorded in macaque prefrontal cortex and $100$ ms bins [18]). (a): Entropy of the $C_{\theta_1,\theta_2,\omega}$ based models. (b): Difference between the entropy of the $C_{\theta_1,\theta_2,\omega}$-based models and the model with independent elements in percent of the independent model.

Fig. 2(a) shows the entropy of the Frank shuffle-based models with Poisson and negative binomial marginals for uncorrelated but dependent elements. $\theta_1$ was varied while $\theta_2$ was estimated using the line-search algorithm for constrained nonlinear minimization [23] with the absolute correlation coefficient as the objective function. Independence is attained for $\theta_1 = 0$. With increasing dependence the entropy decreases until it reaches a minimum at $\theta_1 = -20$. Afterward, it increases again. This is due to the shape of the marginal distributions. The region of strong dependence shifts to a region with small mass. Therefore, the actual dependence decreases. However, in this region the dependency is almost deterministic and thus does not represent a relevant case.

Fig. 2(b) shows the difference to the entropy of corresponding models with independent elements. The entropy deviated by up to $25$ % for the Poisson marginals and up to $15$ % for the negative binomial marginals. So the entropy varies indeed considerably in spite of fixed marginals and un-correlated elements.

We constructed a copula family which allowed us to vary the dependence strength systematically while keeping the variables uncorrelated. It could be argued that this is a pathological example. In the next section, however, we show that such effects can occur even in simple biologically realistic network models.

### 3.2 LIF network

We will now explore the feasibility of uncorrelated spike counts with strong dependencies in a biologically realistic network model. For this purpose, we set up a network of leaky integrate-and-fire neurons (see Section 2.3). The neurons have two common input populations which introduce opposite dependencies (see Fig. 3(a)). Therefore, the correlation should vanish for the right proportion of input strengths. Note that the bottom input population does not contradict to Dale's principle, since excitatory neurons can project to both excitatory and inhibitory neurons.

We can find a copula family which can model this relation and has two separate parameters for the strengths of the input populations:

$$
\begin{aligned}
C_{\theta_1,\theta_2}^{cm}(u,v) = &\frac{1}{2}\left(\max\left\{u^{-\theta_1} + v^{-\theta_1} - 1, 0\right\}\right)^{-1/\theta_1} \\
&+ \frac{1}{2}\left(u - \left(\max\left\{u^{-\theta_2} + (1-v)^{-\theta_2} - 1, 0\right\}\right)^{-1/\theta_2}\right),
\end{aligned} \tag{2}
$$

where $\theta_1, \theta_2 \in (0,\infty)$. It is a mixture of the well known Clayton copula and an one element survival transformation of the Clayton copula [16]. As a mixture of copulas this function is again a copula. A copula of this family is shown in Fig. 3(b).

Fig. 3(c) shows the correlation coefficients of the network generated spike counts and of $C_{\theta_1,\theta_2}^{cm}$ fits. The rate of population D that introduces negative dependence is kept constant, while the rate of population B that introduces positive dependence is varied. The resulting spike count statistics

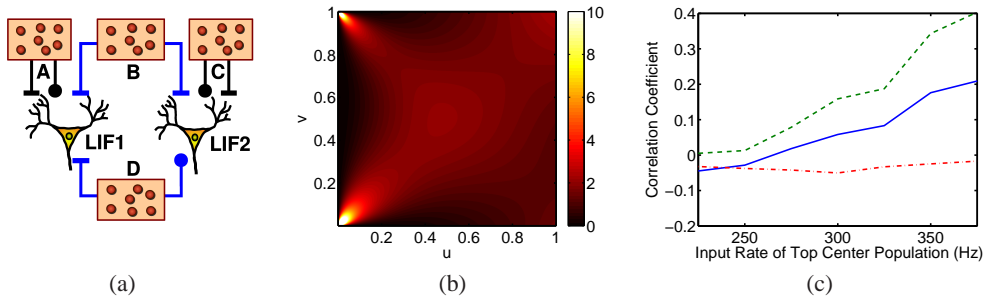

(a)            (b)            (c)

Figure 3: Strong dependence with zero correlation in a biological network model. (a): Neural network models used to generate synthetic spike count data. Two leaky integrate-and-fire neurons (LIF1 and LIF2, see Section 2.3) receive spike inputs (circles for excitation, bars for inhibition) from four separate populations of neurons (rectangular boxes and circles, A-D), but only two populations (B, D) send input to both neurons. All input spike trains were Poisson-distributed. (b): Probability density of the Clayton mixture model $C^{cm}_{\theta_1,\theta_2}$ with $\theta_1 = 1.5$ and $\theta_2 = 2.0$. (c): Correlation coefficients of network generated spike counts compared to correlations of a maximum likelihood fit of the $C^{cm}_{\theta_1,\theta_2}$ copula family to these counts. Solid line: correlation coefficients of counts generated by the network shown in (a). Each neuron had a total inhibitory input rate of $300\,\mathrm{Hz}$ and a total excitatory input rate of $900\,\mathrm{Hz}$. Population D had a rate of $150\,\mathrm{Hz}$. We increased the absolute correlation between the spike counts by shifting the rates: we decreased the rates of A and C and increased the rate of B. The total simulation time amounted to $200\,\mathrm{s}$. Spike counts were calculated for $100\,\mathrm{ms}$ bins. Dashed line: Correlation coefficients of the first mixture component of $C^{cm}_{\theta_1,\theta_2}$. Dashed-dotted line: Correlation coefficients of the second mixture component of $C^{cm}_{\theta_1,\theta_2}$.

were close to typically recorded data. At approximately $275\,\mathrm{Hz}$ the dependencies cancel each other out in the correlation coefficient. Nevertheless, the mixture components of the copula reveal that there are still dependencies: the correlation coefficient of the first mixture component that models negative dependence is relatively constant, while the correlation coefficient of the second mixture component increases with the rate of the corresponding input population. Therefore, correlation coefficients of spike counts that do not at all reflect the true strength of dependence are very likely to occur in biological networks. Structures similar to the investigated network can be formed in any feed-forward network that contains positive and negative weights.

Typically, the network structure is unknown. Hence, it is hard to construct an appropriate copula that is parametrized such that individual dependence strengths are revealed. The goal of the next section is to assess a test that reveals whether the linear correlation coefficient provides an appropriate measure for the dependence.

## 4   Linear correlation test

We will now describe a test for bivariate distributions with Poisson-like marginals that determines whether the dependence structure is well characterized by the linear correlation coefficient. This test combines a variant of the $\chi^2$ goodness-of-fit test for discrete multivariate data with a semiparametric model of linear dependence. We fit the semiparametric model to the data and we apply the goodness-of-fit test to see if the model is adequate for the data.

The semiparametric model that we use consists of the empirical marginals of the sample coupled by a parametric copula family. A dependence structure is well characterized by the linear correlation coefficient if it is Gauss-like. So one way to test for linear dependence would be to use the Gaussian copula family. However, the likelihood of copula-based models relies on the CDF which has no closed form solution for the Gaussian family. Fortunately, a whole class of copula families that are Gauss-like exists. The Frank family is in this class [24] and its CDF can be computed very efficiently. We therefore selected this family for our test (see Eq. 1). The Frank copula has a scalar parameter $\theta$. The parameter relates directly to the dependence. With growing $\theta$ the dependence increases strictly

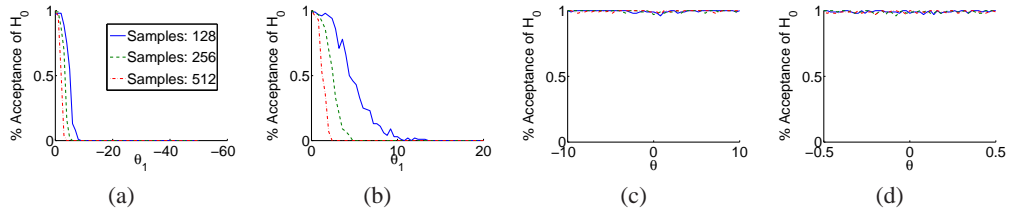

(a)  (b)  (c)  (d)

Figure 4: Percent acceptance of the linear correlation hypothesis for different copula-based models with different dependence strengths and Poisson marginals with rates $\lambda_1 = \lambda_2 = 5$. We used $100$ repetitions each. The number of samples was varied between $128$ and $512$. On the x-axis we varied the strength of the dependence by means of the copula parameters. (a): Frank shuffle family with correlation kept at zero. (b): Clayton mixture family $C^{cm}_{\theta_1,\theta_2}$ with $\theta_1 = 2\theta_2$. (c): Frank family. (d): Gaussian family.

monotonically. For $\theta = 0$ the Frank copula corresponds to independence. Therefore, the usual $\chi^2$ independence test is a special case of our linear correlation test.

The parameter $\theta$ of the Frank family can be estimated based on a maximum likelihood fit. However, this is time-consuming. As an alternative we propose to estimate the copula parameter $\theta$ by means of Kendall's $\tau$. Kendall's $\tau$ is a measure of dependence defined as $\tau(\vec{x}, \vec{y}) = \frac{c-d}{c+d}$, where $c$ is the number of elements in the set $\{(i,j)|(x_i < x_j$ and $y_i < y_j)$ or $(x_i > x_j$ and $y_i > y_j)\}$ and $d$ is the number of element in the set $\{(i,j)|(x_i < x_j$ and $y_i > y_j)$ or $(x_i > x_j$ and $y_i < y_j)\}$ [16]. For the Frank copula with continuous marginals the relation between $\tau$ and $\theta$ is given by $\tau_\theta = 1 - \frac{4}{\theta}[1 - D_1(\theta)]$, where $D_k(x)$ is the Debye function $D_k(x) = \frac{k}{x^k}\int_0^x \frac{t^k}{\exp(t)-1}dt$ [25]. For discrete marginals this is an approximate relation. Unfortunately, $\tau_\theta^{-1}$ cannot be expressed in closed form, but can be easily obtained numerically using Newton's method.

The goodness-of-fit test that we apply for this model is based on the $\chi^2$ test [26]. It is widely applied for testing goodness-of-fit or independence of categorical variables. For the test, observed frequencies are compared to expected frequencies using the following statistic:

$$X^2 = \sum_{i=1}^{k} \frac{(n_i - m_{0i})^2}{m_{0i}}, \tag{3}$$

where $n_i$ are the observed frequencies, $m_{oi}$ are the expected frequencies, and $k$ is the number of bins. For a 2-dimensional table the sum is over both indices of the table. If the frequencies are large enough then $X^2$ is approximately $\chi^2$-distributed with $df = (N-1)(M-1) - s$ degrees of freedom, where $N$ is the number of rows, $M$ is the number of columns, and s is the number of parameters in the $H_0$ model (1 for the Frank family). Thus, for a given significance level $\alpha$ the test accepts the hypothesis $H_0$ that the observed frequencies are a sample from the distribution formed by the expected frequencies, if $X^2$ is less than the $(1 - \alpha)$ point of the $\chi^2$-distribution with $df$ degrees of freedom.

The $\chi^2$ statistic is an asymptotic statistic. In order to be of any value, the frequencies in each bin must be large enough. As a rule of thumb, each frequency should be at least $5$ [26]. This cannot be accomplished for Poisson-like marginals since there is an infinite number of bins. For such cases Loukas and Kemp [27] propose the ordered expected-frequencies procedure. The expected frequencies $m_0$ are sorted monotonically decreasing into a 1-dimensional array. The corresponding observed frequencies form another 1-dimensional array. Then the frequencies in both arrays are grouped from left to right such that the grouped $m_0$ frequencies reach a specified minimum expected frequency (MEF), e.g. MEF= 1 as in [27]. The $\chi^2$ statistic is then estimated using Eq. 3 with the grouped expected and grouped observed frequencies.

To verify the test we applied it to samples from copula-based distributions with Poisson marginals and four different copula families: the Frank shuffle family (Proposition 1), the Clayton mixture family (Eq. 2), the Frank family (Eq. 1), and the Gaussian family (Section 2.1). For the Frank family and the Gaussian family the linear correlation coefficient is well suited to characterize their

dependence. We therefore expected that the test should accept $H_0$, regardless of the dependence strength. In contrast, for the Frank shuffle family and the Clayton mixture family the linear correlation does not reflect the dependence strength. Hence, the test should reject $H_0$ most of the time when there is dependence.

The acceptance rates for these copulas are shown in Fig. 4. For each of the families there was no dependence when the first copula parameter was equal to zero. The Frank and the Gaussian families have only Gauss-like dependence, meaning the correlation coefficient is well-suited to describe the data. In all of these cases the achieved Type I error was small, i.e. the acceptance rate of $H_0$ was close to the desired value (0.95). The plots in (a) and (b) indicate the Type II errors: $H_0$ was accepted although the dependence structure of the counts was not Gauss-like. The Type II error decreased for increasing sample sizes. This is reasonable since $X^2$ is only asymptotically $\chi^2$-distributed. Therefore, the test is unreliable when dependencies and sample sizes are both very small.

## 5 Conclusion

We investigated a worst-case scenario for reliance on the linear correlation coefficient for analyzing dependent spike counts using the Shannon information. The spike counts were uncorrelated but had a strong dependence. Thus, relying solely on the correlation coefficient would lead to an oversight of such dependencies. Although uncorrelated with fixed marginals the information varied by more than 25 %. Therefore, the dependence was not negligible in terms of the entropy. Furthermore, we could show that similar scenarios are very likely to occur in real biological networks. Our test provides a convenient tool to verify whether the correlation coefficient is the right measure for an assessment of the dependence. If the test rejects the Gauss-like dependence hypothesis, more elaborate measures of the dependence should be applied. An adequate copula family provides one way to find such a measure. In general, however, it is hard to find the right parametric family. Directions for future research include a systematic approach for handling the alternative case when one has to deal with the full dependence structure and a closer look at experimentally observed dependencies.

**Acknowledgments.** This work was supported by BMBF grant 01GQ0410.

## A Proof of proposition 1

*Proof.* We show that $C_{\theta_1,\theta_2,\omega}$ is a copula. Since $C_{\theta_1,\theta_2,\omega}$ is commutative we assume w.l.o.g. $u \leq v$. For $u = 0$ or $v = 0$ and for $u = 1$ or $v = 1$ we have $C_{\theta_1,\theta_2,\omega}(u,v) = C_{\theta_1}(u,v)$. Hence, property 1 follows directly from $C_{\theta_1}$. It remains to show that $C_{\theta_1,\theta_2,\omega}$ is 2-increasing (property 2). We will show this in two steps:

1) We show that $C_{\theta_1,\theta_2,\omega}$ is continuous: For $\omega_2 = 1 - \omega$ and $u \in (\omega, \omega_2)$:

$$\lim_{t \nearrow \omega_2} C_{\theta_1,\theta_2,\omega}(u,t) = C_{\theta_1}(u,\omega_2) - \varsigma_{\theta_1}(\omega,\omega,u,\omega_2) + \frac{\varsigma_{\theta_1}(\omega,\omega,u,\omega_2)}{\varsigma_{\theta_2}(\omega,\omega,u,\omega_2)} \varsigma_{\theta_2}(\omega,\omega,u,\omega_2)$$

$$= C_{\theta_1}(u,\omega_2).$$

For $v \in (\omega, 1 - \omega)$:

$$\lim_{t \searrow \omega} C_{\theta_1,\theta_2,\omega}(t,v) = C_{\theta_1}(\omega,v) - \varsigma_{\theta_1}(\omega,\omega,\omega,v) + \lim_{t \searrow \omega} \frac{\varsigma_{\theta_1}(\omega,\omega,t,1-\omega)}{\varsigma_{\theta_2}(\omega,\omega,t,1-\omega)} \varsigma_{\theta_2}(\omega,\omega,t,v).$$

We can use l'Hôpital's rule since $\lim_{t \searrow \omega} \varsigma_\theta(\omega,\omega,t,1-\omega) = 0$. It is easy to verify that

$$\frac{\partial C_\theta}{\partial u}(v) = \frac{e^{-\theta u}(e^{-\theta v} - 1)}{e^{-\theta} - 1 + (e^{-\theta u} - 1)(e^{-\theta v} - 1)}.$$

Thus, the quotient is constant and $\lim_{t \searrow \omega} C_{\theta_1,\theta_2,\omega}(t,v) = C_{\theta_1}(\omega,v) - 0 + 0$.

2) $C_{\theta_1,\theta_2,\omega}$ has non-negative density almost everywhere on $[0,1]^2$. This is obvious for $u_1, v_1 \notin [\omega, 1-\omega]^2$, because $C_{\theta_1}$ is a copula. Straightforward but tedious algebra shows that $\forall u_1, v_1 \in (\omega, 1-\omega)^2 : \frac{\partial^2 C_{\theta_1,\theta_2,\omega}}{\partial u \partial v}(u_1,v_1) \geq 0$.

Thus, $C_{\theta_1,\theta_2,\omega}$ is continuous and has density almost everywhere on $[0,1]^2$ and is therefore 2-increasing. □

# References

[1] M. Jazayeri and J. A. Movshon. Optimal representation of sensory information by neural populations. *Nature Neuroscience*, 9(5):690–696, 2006.

[2] L. Schwabe and K. Obermayer. Adaptivity of tuning functions in a generic recurrent network model of a cortical hypercolumn. *Journal of Neuroscience*, 25(13):3323–3332, 2005.

[3] D. A. Gutnisky and V. Dragoi. Adaptive coding of visual information in neural populations. *Nature*, 452(7184):220–224, 2008.

[4] M. Shamir and H. Sompolinsky. Implications of neuronal diversity on population coding. *Neural Computation*, 18(8):1951–1986, 2006.

[5] P. Series, P. E. Latham, and A. Pouget. Tuning curve sharpening for orientation selectivity: coding efficiency and the impact of correlations. *Nature Neuroscience*, 7(10):1129–1135, 2004.

[6] L. F. Abbott and P. Dayan. The effect of correlated variability on the accuracy of a population code. *Neural Computation*, 11(1):91–101, 1999.

[7] B. B. Averbeck, P. E. Latham, and A. Pouget. Neural correlations, population coding and computation. *Nature Review Neuroscience*, 7(5):358–366, 2006.

[8] Y. Roudi, S. Nirenberg, and P. E. Latham. Pairwise maximum entropy models for studying large biological systems: When they can work and when they can't. *PLoS Computational Biology*, 5(5):e1000380+, 2009.

[9] E. Schneidman, M. J. Berry II, R. Segev, and W. Bialek. Weak pairwise correlations imply strongly correlated network states in a neural population. *Nature*, 440:1007–1012, 2006.

[10] J. Shlens, G. D. Field, J. L. Gauthier, M. I. Grivich, D. Petrusca, E. Sher, A. M. Litke, and E. J. Chichilnisky. The structure of multi-neuron firing patterns in primate retina. *Journal of Neuroscience*, 26:2006, 2006.

[11] B. B. Averbeck and D. Lee. Neural noise and movement-related codes in the macaque supplementary motor area. *Journal of Neuroscience*, 23(20):7630–7641, 2003.

[12] S. Panzeri, G. Pola, F. Petroni, M. P. Young, and R. S. Petersen. A critical assessment of different measures of the information carried by correlated neuronal firing. *Biosystems*, 67(1-3):177–185, 2002.

[13] H. Sompolinsky, H. Yoon, K. Kang, and M. Shamir. Population coding in neuronal systems with correlated noise. *Physical Review E*, 64(5):051904, 2001.

[14] A. Kohn and M. A. Smith. Stimulus dependence of neuronal correlation in primary visual cortex of the macaque. *Journal of Neuroscience*, 25(14):3661–3673, 2005.

[15] W. Bair, E. Zohary, and W. T. Newsome. Correlated firing in macaque visual area MT: time scales and relationship to behavior. *Journal of Neuroscience*, 21(5):1676–1697, 2001.

[16] R. B. Nelsen. *An Introduction to Copulas*. Springer, New York, second edition, 2006.

[17] M. J. Frank. On the simultaneous associativity of f(x,y) and x+y-f(x,y). *Aequations Math*, 19:194–226, 1979.

[18] A. Onken, S. Grünewälder, M. Munk, and K. Obermayer. Modeling short-term noise dependence of spike counts in macaque prefrontal cortex. In D. Koller, D. Schuurmans, Y. Bengio, and L. Bottou, editors, *Advances in Neural Information Processing Systems 21*, pages 1233–1240, 2009.

[19] P. Berkes, F. Wood, and J. Pillow. Characterizing neural dependencies with copula models. In D. Koller, D. Schuurmans, Y. Bengio, and L. Bottou, editors, *Advances in Neural Information Processing Systems 21*, pages 129–136, 2009.

[20] D. J. Tolhurst, J. A. Movshon, and A. F. Dean. The statistical reliability of signals in single neurons in cat and monkey visual cortex. *Vision Research*, 23:775–785, 1982.

[21] C. E. Shannon. A mathematical theory of communication. *Bell System Technical Journal*, 27:379–423, 1948.

[22] P. Dayan and L. F. Abbott. *Theoretical Neuroscience*. Cambridge (Massachusetts): MIT Press, 2001.

[23] R. A. Waltz, J. L. Morales, J. Nocedal, and D. Orban. An interior algorithm for nonlinear optimization that combines line search and trust region steps. *Mathematical Programming*, 107(3):391–408, 2006.

[24] C. Genest, B. Rémillard, and D. Beaudoin. Goodness-of-fit tests for copulas: A review and a power study. *Insurance: Mathematics and Economics*, 44(2):199–213, 2009.

[25] C. Genest. Frank's family of bivariate distributions. *Biometrika*, 74:549–555, 1987.

[26] W. G. Cochran. The $\chi^2$ test of goodness of fit. *Annals of Mathematical Statistics*, 23(3):315–345, 1952.

[27] S. Loukas and C. D. Kemp. On the chi-square goodness-of-fit statistic for bivariate discrete distributions. *The Statistician*, 35:525–529, 1986.
